# Learning in large linear perceptrons and why the thermodynamic limit is relevant to the real world

Peter Sollich
Department of Physics, University of Edinburgh
Edinburgh EH9 3JZ, U.K.
P.Sollich@ed.ac.uk

## Abstract

We present a new method for obtaining the response function $\mathcal{G}$ and its average $G$ from which most of the properties of learning and generalization in linear perceptrons can be derived. We first rederive the known results for the 'thermodynamic limit' of infinite perceptron size $N$ and show explicitly that $\mathcal{G}$ is self-averaging in this limit. We then discuss extensions of our method to more general learning scenarios with anisotropic teacher space priors, input distributions, and weight decay terms. Finally, we use our method to calculate the finite $N$ corrections of order $1/N$ to $G$ and discuss the corresponding finite size effects on generalization and learning dynamics. An important spin-off is the observation that results obtained in the thermodynamic limit are often directly relevant to systems of fairly modest, 'real-world' sizes.

## 1 INTRODUCTION

One of the main areas of research within the Neural Networks community is the issue of learning and generalization. Starting from a set of training examples (normally assumed to be input-output pairs) generated by some unknown 'teacher' rule $\mathcal{V}$, one wants to find, using a suitable learning or training algorithm, a student $\mathcal{N}$ (read 'Neural Network') which generalizes from the training set, $i.e.$, predicts the outputs corresponding to inputs not contained in the training set as accurately as possible.

If the inputs are $N$-dimensional vectors $\mathbf{x} \in \mathcal{R}^N$ and the outputs are scalars $y \in \mathcal{R}$, then one of the simplest functional forms that can be assumed for the student $\mathcal{N}$ is the linear perceptron, which is parametrized in terms of a weight vector $\mathbf{w}_\mathcal{N} \in \mathcal{R}^N$ and implements the linear input-output mapping

$$y_\mathcal{N}(\mathbf{x}) = \tfrac{1}{\sqrt{N}} \mathbf{w}_\mathcal{N}^T \mathbf{x}. \tag{1}$$

A commonly used learning algorithm for the linear perceptron is gradient descent on the training error, *i.e.*, the error that the student $\mathcal{N}$ makes on the training set. Using the standard squared output deviation error measure, the training error for a given set of $p$ training examples $\{(\mathbf{x}^\mu, y^\mu), \mu = 1 \ldots p\}$ is $E_t = \sum_\mu \tfrac{1}{2}(y^\mu - y_\mathcal{N}(\mathbf{x}^\mu))^2 = \tfrac{1}{2}\sum_\mu (y^\mu - \mathbf{w}_\mathcal{N}^T \mathbf{x}^\mu/\sqrt{N})^2$. To prevent the student from fitting noise in the training data, a quadratic weight decay term $\tfrac{1}{2}\lambda \mathbf{w}_\mathcal{N}^2$ is normally added to the training error, with the value of the weight decay parameter $\lambda$ determining how strongly large weight vectors are penalized. Gradient descent is thus performed on the function $E = E_t + \tfrac{1}{2}\lambda \mathbf{w}_\mathcal{N}^2$, and the corresponding learning dynamics is, in a continuous time approximation, $d\mathbf{w}_\mathcal{N}/dt = -\nabla_\mathbf{w} E$. As discussed in detail by Krogh and Hertz (1992), this results in an exponential approach of $\mathbf{w}_\mathcal{N}$ to its asymptotic value, with decay constants given by the eigenvalues of the matrix $\mathbf{M}_\mathcal{N}$, defined by ($\mathbf{1}$ denotes the $N \times N$ identity matrix)

$$\mathbf{M}_\mathcal{N} = \lambda \mathbf{1} + \mathbf{A}, \quad \mathbf{A} = \tfrac{1}{N}\sum_\mu \mathbf{x}^\mu (\mathbf{x}^\mu)^T.$$

To examine what generalization performance is achieved by the above learning algorithm, one has to make an assumption about the functional form of the teacher. The simplest such assumption is that the problem is learnable, *i.e.*, that the teacher, like the student, is a linear perceptron. A teacher $\mathcal{V}$ is then specified by a weight vector $\mathbf{w}_\mathcal{V}$ and maps a given input $\mathbf{x}$ to the output $y_\mathcal{V}(\mathbf{x}) = \mathbf{w}_\mathcal{V}^T \mathbf{x}/\sqrt{N}$. We assume that the test inputs for which the student is asked to predict the corresponding outputs are drawn from an isotropic Gaussian distribution, $P(\mathbf{x}) \propto \exp(-\tfrac{1}{2}\mathbf{x}^2)$. The generalization error, *i.e.*, the average error that a student $\mathcal{N}$ makes on a random input when compared to teacher $\mathcal{V}$, is given by

$$\epsilon_\mathrm{g} = \tfrac{1}{2}\langle (y_\mathcal{N}(\mathbf{x}) - y_\mathcal{V}(\mathbf{x}))^2 \rangle_{P(\mathbf{x})} = \tfrac{1}{2N}(\mathbf{w}_\mathcal{N} - \mathbf{w}_\mathcal{V})^2. \tag{2}$$

Inserting the learning dynamics $\mathbf{w}_\mathcal{N} = \mathbf{w}_\mathcal{N}(t)$, the generalization acquires a time dependence, which in its exact form depends on the specific training set, teacher, and initial value of the student weight vector, $\mathbf{w}_\mathcal{N}(t = 0)$. We shall confine our attention to the average of this time-dependent generalization error over all possible training sets and teachers; to avoid clutter, we write this average simply as $\epsilon_\mathrm{g}(t)$. We assume that the inputs $\mathbf{x}^\mu$ in the training set are chosen independently and randomly from the same distribution as the test inputs, and that the corresponding training outputs are the teacher outputs corrupted by additive noise, $y^\mu = y_\mathcal{V}(\mathbf{x}^\mu) + \eta^\mu$, where the $\eta^\mu$ have zero mean and variance $\sigma^2$. If we further assume an isotropic Gaussian prior on the teacher weight vectors, $P(\mathbf{w}_\mathcal{V}) \propto \exp(-\tfrac{1}{2}\mathbf{w}_\mathcal{V}^2)$, then the average generalization error for $t \to \infty$ is (Krogh and Hertz, 1992)

$$\epsilon_\mathrm{g}(t \to \infty) = \frac{1}{2}\left[ \sigma^2 G + \lambda(\sigma^2 - \lambda)\frac{\partial G}{\partial \lambda} \right], \tag{3}$$

where $G$ is the average of the so-called *response function* over the training inputs:

$$G = \langle \mathcal{G} \rangle_{P(\{\mathbf{x}^\mu\})}, \quad \mathcal{G} = \tfrac{1}{N}\mathrm{tr}\, \mathbf{M}_\mathcal{N}^{-1}. \tag{4}$$

The time dependence of the average generalization error for finite but large $t$ is an exponential approach to the asymptotic value (3) with decay constant $\lambda + a_{\min}$, where $a_{\min}$ is the lowest eigenvalue occurring in the average eigenvalue spectrum of the input correlation matrix $\mathbf{A}$ (Krogh and Hertz, 1992). This average eigenvalue spectrum, which we denote by $\rho(a)$, can be calculated from the average response function according to (Krogh, 1992)

$$\rho(a) = \frac{1}{\pi} \lim_{\epsilon \to 0+} \operatorname{Im} G|_{\lambda = -a - i\epsilon}, \tag{5}$$

where we have assumed $\rho(a)$ to be normalized, $\int da\, \rho(a) = 1$.

Eqs. (3,5) show that the key quantity determining learning and generalization in the linear perceptron is the average response function $G$ defined in (4). This function has previously been calculated in the 'thermodynamic limit', $N \to \infty$ at $\alpha = p/N = \text{const.}$, using a diagrammatic expansion (Hertz *et al.*, 1989) and the replica method (Opper, 1989, Kinzel and Opper, 1991). In Section 2, we present what we believe to be a much simpler method for calculating $G$, based only on simple matrix identities. We also show explicitly that $\mathcal{G}$ is self-averaging in the thermodynamic limit, which means that the fluctuations of $\mathcal{G}$ around its average $G$ become vanishingly small as $N \to \infty$. This implies, for example, that the generalization error is also self-averaging. In Section 3 we extend the method to more general cases such as anisotropic teacher space priors and input distributions, and general quadratic penalty terms. Finite size effects are considered in Section 4, where we calculate the $O(1/N)$ corrections to $G$, $\epsilon_g(t \to \infty)$ and $\rho(a)$. We discuss the resulting effects on generalization and learning dynamics and derive explicit conditions on the perceptron size $N$ for results obtained in the thermodynamic limit to be valid. We conclude in Section 5 with a brief summary and discussion of our results.

## 2 THE BASIC METHOD

Our method for calculating the average response function $G$ is based on a recursion relation relating the values of the (unaveraged) response function $\mathcal{G}$ for $p$ and $p+1$ training examples. Assume that we are given a set of $p$ training examples with corresponding matrix $\mathbf{M}_N$. By adding a new training example with input $\mathbf{x}$, we obtain the matrix $\mathbf{M}_N^+ = \mathbf{M}_N + \frac{1}{N}\mathbf{x}\mathbf{x}^T$. It is straightforward to show that the inverse of $\mathbf{M}_N^+$ can be expressed as

$$\left(\mathbf{M}_N^+\right)^{-1} = \mathbf{M}_N^{-1} - \frac{\frac{1}{N}\mathbf{M}_N^{-1}\mathbf{x}\mathbf{x}^T\mathbf{M}_N^{-1}}{1 + \frac{1}{N}\mathbf{x}^T\mathbf{M}_N^{-1}\mathbf{x}}.$$

(One way of proving this identity is to multiply both sides by $\mathbf{M}_N^+$ and exploit the fact that $\mathbf{M}_N^+\mathbf{M}_N^{-1} = 1 + \frac{1}{N}\mathbf{x}\mathbf{x}^T\mathbf{M}_N^{-1}$.) Taking the trace, we obtain the following recursion relation for $\mathcal{G}$:

$$\mathcal{G}(p+1) = \mathcal{G}(p) - \frac{1}{N} \frac{\frac{1}{N}\mathbf{x}^T\mathbf{M}_N^{-2}\mathbf{x}}{1 + \frac{1}{N}\mathbf{x}^T\mathbf{M}_N^{-1}\mathbf{x}}. \tag{6}$$

Now denote $z_i = \frac{1}{N}\mathbf{x}^T\mathbf{M}_N^{-i}\mathbf{x}$ ($i = 1, 2$). With $\mathbf{x}$ drawn randomly from the assumed input distribution $P(\mathbf{x}) \propto \exp(-\frac{1}{2}\mathbf{x}^2)$, the $z_i$ can readily be shown to be random

variables with means and (co-)variances

$$\langle z_i \rangle = \frac{1}{N} \operatorname{tr} \mathbf{M}_N^{-i}, \quad \langle \Delta z_i \Delta z_j \rangle = \frac{2}{N^2} \operatorname{tr} \mathbf{M}_N^{-i-j}.$$

Combining this with the fact that $\operatorname{tr} \mathbf{M}_N^{-k} \leq N \lambda^{-k} = O(N)$, we have that the fluctuations $\Delta z_i$ of the $z_i$ around their average values are $O(1/\sqrt{N})$; inserting this into (6), we obtain

$$
\begin{aligned}
\mathcal{G}(p+1) &= \mathcal{G}(p) - \frac{1}{N} \frac{\frac{1}{N} \operatorname{tr} \mathbf{M}_N^{-2}}{1 + \frac{1}{N} \operatorname{tr} \mathbf{M}_N^{-1}} + O(N^{-3/2}) \\
&= \mathcal{G}(p) + \frac{1}{N} \frac{\partial \mathcal{G}(p)}{\partial \lambda} \frac{1}{1 + \mathcal{G}(p)} + O(N^{-3/2}).
\end{aligned}
\tag{7}
$$

Starting from $\mathcal{G}(0) = 1/\lambda$, we can apply this recursion $p = \alpha N$ times to obtain $\mathcal{G}(p)$ up to terms which add up to at most $O(pN^{-3/2}) = O(1/\sqrt{N})$. This shows that $\mathcal{G}$ is self-averaging in the thermodynamic limit: whatever the training set, the value of $\mathcal{G}$ will always be the same up to fluctuations of $O(1/\sqrt{N})$. In fact, we shall show in Section 4 that the fluctuations of $\mathcal{G}$ are only $O(1/N)$. This means that the $O(N^{-3/2})$ fluctuations from each iteration of (7) are only weakly correlated, so that they add up like independent random variables to give a total fluctuation for $\mathcal{G}(p)$ of $O((p/N^3)^{1/2}) = O(1/N)$.

We have seen that, in the thermodynamic limit, $\mathcal{G}$ is identical to its average $G$ because its fluctuations are vanishingly small. To calculate the value of $G$ in the thermodynamic limit as a function of $\alpha$ and $\lambda$, we insert the relation $\mathcal{G}(p+1) - \mathcal{G}(p) = \frac{1}{N} \partial \mathcal{G}(\alpha)/\partial \alpha + O(1/N^2)$ into eq. (7) (with $\mathcal{G}$ replaced by $G$) and neglect all finite $N$ corrections. This yields the partial differential equation

$$\frac{\partial G}{\partial \alpha} - \frac{\partial G}{\partial \lambda} \frac{1}{1 + G} = 0, \tag{8}$$

which can readily be solved using the method of characteristic curves (see, *e.g.*, John, 1978). Using the initial condition $G|_{\alpha=0} = 1/\lambda$ gives $\alpha/(1 + G) = 1/G - \lambda$, which leads to the well-known result (see, *e.g.*, Hertz *et al.*, 1989)

$$G = \frac{1}{2\lambda} \left( 1 - \alpha - \lambda + \sqrt{(1 - \alpha - \lambda)^2 + 4\lambda} \right). \tag{9}$$

In the complex $\lambda$ plane, $G$ has a pole at $\lambda = 0$ and a branch cut arising from the root; according to eq. (5), these singularities determine the average eigenvalue spectrum $\rho(a)$ of $\mathbf{A}$, with the result (Krogh, 1992)

$$\rho(a) = (1 - \alpha)\Theta(1 - \alpha)\delta(a) + \frac{1}{2\pi a} \sqrt{(a_+ - a)(a - a_-)}, \tag{10}$$

where $\Theta(x)$ is the Heaviside step function, $\Theta(x) = 1$ for $x > 0$ and 0 otherwise. The root in eq. (10) only contributes when its argument is non-negative, *i.e.*, for $a$ between the 'spectral limits' $a_-$ and $a_+$, which have the values $a_\pm = (1 \pm \sqrt{\alpha})^2$.

## 3   EXTENSIONS TO MORE GENERAL LEARNING SCENARIOS

We now discuss some extensions of our method to more general learning scenarios. First, consider the case of an anisotropic teacher space prior, $P(\mathbf{w}_\nu) \propto$

$\exp(-\frac{1}{2}\mathbf{w}_\nu^T \Sigma_\nu^{-1} \mathbf{w}_\nu)$, with symmetric positive definite $\Sigma_\nu$. This leaves the definition of the response function unchanged; eq. (3), however, has to be replaced by $\epsilon_g(t \to \infty) = 1/2\{\sigma^2 G + \lambda[\sigma^2 - \lambda(\frac{1}{N}\text{tr } \Sigma_\nu)]\partial G/\partial \lambda\}$.

As a second extension, assume that the inputs are drawn from an anisotropic distribution, $P(\mathbf{x}) \propto \exp(-\frac{1}{2}\mathbf{x}^T \Sigma^{-1} \mathbf{x})$. It can then be shown that the asymptotic value of the average generalization error is still given by eq. (3) if the response function is redefined to be $\mathcal{G} = \frac{1}{N}\text{tr } \Sigma \mathbf{M}_\mathcal{N}^{-1}$. This modified response function can be calculated as follows: First we rewrite $\mathcal{G}$ as $\frac{1}{N}\text{tr } (\lambda\Sigma^{-1} + \tilde{\mathbf{A}})^{-1}$, where $\tilde{\mathbf{A}} = \frac{1}{N}\sum_\mu(\tilde{\mathbf{x}}^\mu)^T \tilde{\mathbf{x}}^\mu$ is the correlation matrix of the transformed input examples $\tilde{\mathbf{x}}^\mu = \Sigma^{-1/2}\mathbf{x}^\mu$. Since the $\tilde{\mathbf{x}}^\mu$ are distributed according to $P(\tilde{\mathbf{x}}^\mu) \propto \exp(-\frac{1}{2}(\tilde{\mathbf{x}}^\mu)^2)$, the problem is thus reduced to finding the response function $\mathcal{G} = \frac{1}{N}\text{tr } (\mathbf{L} + \mathbf{A})^{-1}$ for isotropically distributed inputs and $\mathbf{L} = \lambda\Sigma^{-1}$. The recursion relations between $\mathcal{G}(p+1)$ and $\mathcal{G}(p)$ derived in the previous section remain valid, and result, in the thermodynamic limit, in a differential equation for the average response function $G$ analogous to eq. (8). The initial condition is now $G|_{\alpha=0} = \frac{1}{N}\text{tr } \mathbf{L}^{-1}$, and one obtains an implicit equation for $G$,

$$G = \frac{1}{N}\text{tr }\left(\mathbf{L} + \frac{\alpha}{1+G}\mathbf{1}\right)^{-1}, \qquad (11)$$

where in the case of an anisotropic input distribution considered here, $\mathbf{L} = \lambda\Sigma^{-1}$. If $\Sigma$ has a particularly simple form, then the dependence of $G$ on $\alpha$ and $\lambda$ can be obtained analytically, but in general eq. (11) has to solved numerically.

Finally, one can also investigate the effect of a general quadratic weight decay term, $\frac{1}{2}\mathbf{w}_\mathcal{N}^T \Lambda \mathbf{w}_\mathcal{N}$, in the energy function $E$. The expression for the average generalization error becomes more cumbersome than eq. (3) in this case, but the result can still be expressed in terms of the average response function $G = \langle\mathcal{G}\rangle = \langle\frac{1}{N}\text{tr } (\Lambda + \mathbf{A})^{-1}\rangle$, which can be obtained as the solution of eq. (11) for $\mathbf{L} = \Lambda$.

## 4   FINITE $N$ CORRECTIONS

So far, we have focussed attention on the thermodynamic limit of perceptrons of infinite size $N$. The results are clearly only approximately valid for real, finite systems, and it is therefore interesting to investigate corrections for finite $N$. This we do in the present section by calculating the $O(1/N)$ corrections to $G$ and $\rho(a)$. For details of the calculations and results of computer simulations which support our theoretical analysis, we refer the reader to (Sollich, 1994).

First note that, for $\lambda = 0$, the exact result for the average response function is $G|_{\lambda=0} = (\alpha - 1 - 1/N)^{-1}$ for $\alpha > 1 + 1/N$ (see, *e.g.*, Eaton, 1983), which clearly admits a series expansion in powers of $1/N$. We assume that a similar expansion also exists for nonzero $\lambda$, and write

$$G = G_0 + G_1/N + O(1/N^2). \qquad (12)$$

$G_0$ is the value of $G$ in the thermodynamic limit as given by eq. (9). For finite $N$, the fluctuations $\Delta\mathcal{G} = \mathcal{G} - G$ of $\mathcal{G}$ around its average value $G$ become relevant; for $\lambda = 0$, the variance of these fluctuations is known to have a power series expansion in $1/N$, and again we assume a similar expansion for finite $\lambda$, $\langle(\Delta\mathcal{G})^2\rangle = \Delta^2/N + O(1/N^2)$,

where the first term is $O(1/N)$ and not $O(1)$ because, as discussed in Section 2, the fluctuations of $\mathcal{G}$ for large $N$ are no greater than $O(1/\sqrt{N})$. To calculate $G_1$ and $\Delta^2$, one starts again from the recursion relation (6), now expanding everything up to second order in the fluctuation quantities $\Delta z_i$ and $\Delta \mathcal{G}$. Averaging over the training inputs and collecting orders of $1/N$ yields after some straightforward algebra the known eq. (8) for $G_0$ and two linear partial differential equations for $G_1$ and $\Delta^2$, the latter obtained by squaring both sides of eq. (6). Solving these, one obtains

$$\Delta^2 \equiv 0, \quad G_1 = \frac{G_0^2(1 - \lambda G_0)}{(1 + \lambda G_0^2)^2}. \tag{13}$$

In the limit $\lambda \to 0$, $G_1 = 1/(\alpha - 1)^2$ consistent with the exact result for $G$ quoted above; likewise, the result $\Delta^2 \equiv 0$ agrees with the exact series expansion of the variance of the fluctuations of $\mathcal{G}$ for $\lambda = 0$, which begins with an $O(1/N^2)$ term (see, *e.g.*, Barber *et al.*, 1994).

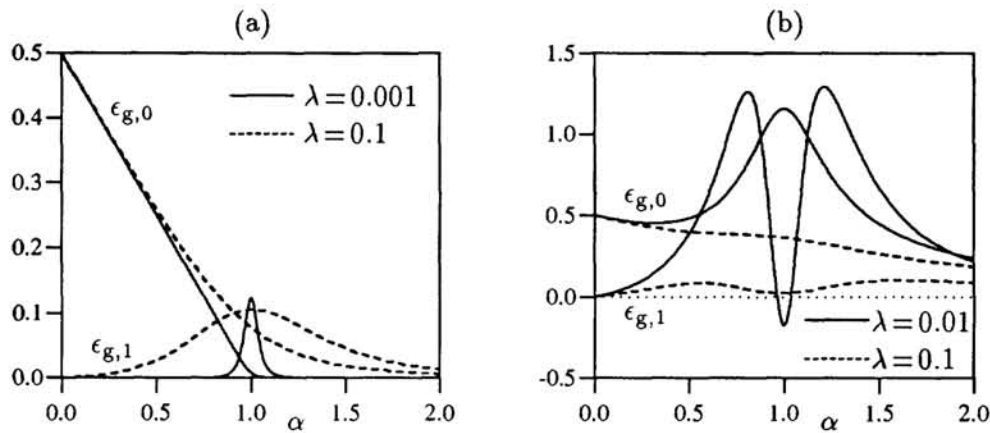

Figure 1: Average generalization error: Result for $N \to \infty$, $\epsilon_{g,0}$, and coefficient of $O(1/N)$ correction, $\epsilon_{g,1}$. (a) Noise free teacher, $\sigma^2 = 0$. (b) Noisy teacher, $\sigma^2 = 0.5$. Curves are labeled by the value of the weight decay parameter $\lambda$.

From the $1/N$ expansion (12) of $G$ we obtain, using eq. (3), a corresponding expansion of the asymptotic value of the average generalization error, which we write as $\epsilon_g(t \to \infty) = \epsilon_{g,0} + \epsilon_{g,1}/N + O(1/N^2)$. It follows that the thermodynamic limit result for the average generalization error, $\epsilon_{g,0}$, is a good approximation to the true result for finite $N$ as long as $N \gg N_c = |\epsilon_{g,1}/\epsilon_{g,0}|$. In Figure 1, we plot $\epsilon_{g,0}$ and $\epsilon_{g,1}$ for several values of $\lambda$ and $\sigma^2$. It can be seen that the relative size of the first order correction $|\epsilon_{g,1}/\epsilon_{g,0}|$ and hence the critical system size $N_c$ for validity of the thermodynamic limit result is largest when $\lambda$ is small. Exploiting this fact, $N_c$ can be bounded by $1/(1 - \alpha)$ for $\alpha < 1$ and $(3\alpha + 1)/[\alpha(\alpha - 1)]$ for $\alpha > 1$. It follows, for example, that the critical system size $N_c$ is smaller than 5 as long as $\alpha < 0.8$ or $\alpha > 1.72$, for all $\lambda$ and $\sigma^2$. This bound on $N_c$ can be tightened for non-zero $\lambda$; for $\lambda > 2$, for example, one has $N_c < (2\lambda - 1)/(\lambda + 1)^2 < 1/3$. We have thus shown explicitly that thermodynamic limit calculations of learning and generalization behaviour can be relevant for fairly small, 'real-world' systems of size $N$ of the order of a few tens or hundreds. This is in contrast to the widespread suspicion

among non-physicists that the methods of statistical physics give valid results only for huge system sizes of the order of $N \approx 10^{23}$.

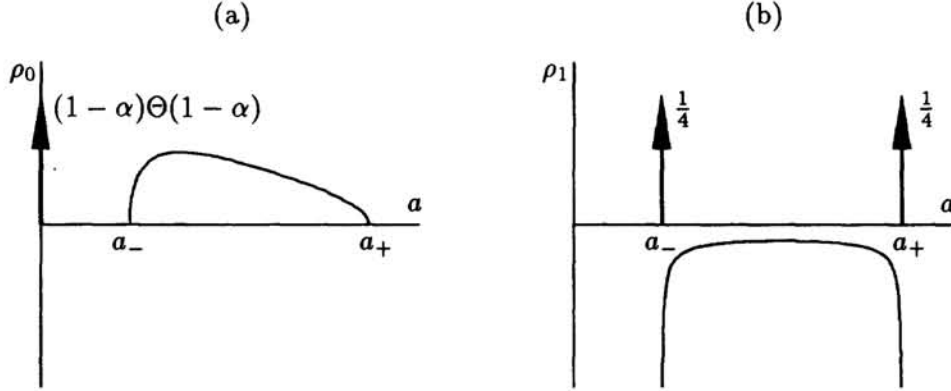

Figure 2: Schematic plot of the average eigenvalue spectrum $\rho(a)$ of the input correlation matrix **A**. (a) Result for $N \to \infty$, $\rho_0(a)$. (b) $O(1/N)$ correction, $\rho_1(a)$. Arrows indicate $\delta$-peaks and are labeled by the corresponding heights.

We now consider the $O(1/N)$ correction to the average eigenvalue spectrum of the input correlation matrix **A**. Setting $\rho(a) = \rho_0(a) + \rho_1(a)/N + O(1/N^2)$, $\rho_0(a)$ is the $N \to \infty$ result given by eq. (10), and from eq. (13) one derives

$$\rho_1(a) = \frac{1}{4}\delta(a - a_+) + \frac{1}{4}\delta(a - a_-) - \frac{1}{2\pi}\frac{1}{\sqrt{(a_+ - a)(a - a_-)}}.$$

Figure 2 shows sketches of $\rho_0(a)$ and $\rho_1(a)$. Note that $\int da\, \rho_1(a) = 0$ as expected since the normalization of $\rho(a)$ is independent of $N$. Furthermore, there is no $O(1/N)$ correction to the $\delta$-peak in $\rho_0(a)$ at $a = 0$, since this peak arises from the $N - p$ zero eigenvalues of **A** for $\alpha = p/N < 1$ and therefore has a height of $1 - \alpha$ for any finite $N$. The $\delta$-peaks in $\rho_1(a)$ at the spectral limits $a_+$ and $a_-$ are an artefact of the truncated $1/N$ expansion: $\rho(a)$ is determined by the singularities of $G$ as a function of $\lambda$, and the location of these singularities is only obtained correctly by resumming the full $1/N$ expansion. The $\delta$-peaks in $\rho_1(a)$ can be interpreted as 'precursors' of a broadening of the eigenvalue spectrum of **A** to values which, when the whole $1/N$ series is resummed, will lie outside the $N \to \infty$ spectral range $[a_-, a_+]$. The negative term in $\rho_1(a)$ represents the corresponding 'flattening' of the eigenvalue spectrum between $a_-$ and $a_+$. We can thus conclude that the average eigenvalue spectrum of **A** for finite $N$ will be broader than for $N \to \infty$, which means in particular that the learning dynamics will be slowed down since the smallest eigenvalue $a_{\min}$ of **A** will be smaller than $a_-$. From our result for $\rho_1(a)$ we can also deduce when the $N \to \infty$ result $\rho_0(a)$ is valid for finite $N$; the condition turns out to be $N \gg a/[(a_+ - a)(a - a_-)]$. Consistent with our discussion of the broadening of the eigenvalue spectrum of **A**, $N$ has to be larger for $a$ near the spectral limits $a_-, a_+$ if $\rho_0(a)$ is to be a good approximation to the finite $N$ average eigenvalue spectrum of **A**.

## 5   SUMMARY AND DISCUSSION

We have presented a new method, based on simple matrix identities, for calculating the response function $\mathcal{G}$ and its average $G$ which determine most of the properties of learning and generalization in linear perceptrons. In the thermodynamic limit, $N \to \infty$, we have recovered the known result for $G$ and have shown explicitly that $\mathcal{G}$ is self-averaging. Extensions of our method to more general learning scenarios have also been discussed. Finally, we have obtained the $O(1/N)$ corrections to $G$ and the corresponding corrections to the average generalization error, and shown explicitly that the results obtained in the thermodynamic limit can be valid for fairly small, 'real-world' system sizes $N$. We have also calculated the $O(1/N)$ correction to the average eigenvalue spectrum of the input correlation matrix $\mathbf{A}$ and interpreted it in terms of a broadening of the spectrum for finite $N$, which will cause a slowing down of the learning dynamics.

We remark that the $O(1/N)$ corrections that we have obtained can also be used in different contexts, for example for calculations of test error fluctuations and optimal test set size (Barber *et al.*, 1994). Another application is in an analysis of the evidence procedure in Bayesian inference for finite $N$, where optimal values of 'hyperparameters' like the weight decay parameter $\lambda$ are determined on the basis of the training data (G Marion, in preparation). We hope, therefore, that our results will pave the way for a systematic investigation of finite size effects in learning and generalization.

### References

D Barber, D Saad, and P Sollich (1994). Finite size effects and optimal test set size in linear perceptrons. Submitted to *J. Phys. A*.

M L Eaton (1983). *Multivariate Statistics - A Vector Space Approach*. Wiley, New York.

J A Hertz, A Krogh, and G I Thorbergsson (1989). Phase transitions in simple learning. *J. Phys. A*, 22:2133–2150.

F John (1978). *Partial Differential Equations*. Springer, New York, 3rd ed.

W Kinzel and M Opper (1991). Dynamics of learning. In E Domany, J L van Hemmen, and K Schulten, editors, *Models of Neural Networks*, pages 149–171. Springer, Berlin.

A Krogh (1992). Learning with noise in a linear perceptron. *J. Phys. A*, 25:1119–1133.

A Krogh and J A Hertz (1992). Generalization in a linear perceptron in the presence of noise. *J. Phys. A*, 25:1135–1147.

M Opper (1989). Learning in neural networks: Solvable dynamics. *Europhysics Letters*, 8:389–392.

P Sollich (1994). Finite-size effects in learning and generalization in linear perceptrons. *J. Phys. A*, 27:7771–7784.
